# Harmonising Chorales by Probabilistic Inference

**Moray Allan and Christopher K. I. Williams**
School of Informatics, University of Edinburgh
Edinburgh EH1 2QL
moray.allan@ed.ac.uk, c.k.i.williams@ed.ac.uk

## Abstract

We describe how we used a data set of chorale harmonisations composed by Johann Sebastian Bach to train Hidden Markov Models. Using a probabilistic framework allows us to create a harmonisation system which learns from examples, and which can compose new harmonisations. We make a quantitative comparison of our system's harmonisation performance against simpler models, and provide example harmonisations.

## 1 Introduction

Chorale harmonisation is a traditional part of the theoretical education of Western classical musicians. Given a melody, the task is to create three further lines of music which will sound pleasant when played simultaneously with the original melody. A good chorale harmonisation will show an understanding of the basic 'rules' of harmonisation, which codify the aesthetic preferences of the style. Here we approach chorale harmonisation as a machine learning task, in a probabilistic framework. We use example harmonisations to build a model of harmonic processes. This model can then be used to compose novel harmonisations.

Section 2 below gives an overview of the musical background to chorale harmonisation. Section 3 explains how we can create a harmonisation system using Hidden Markov Models. Section 4 examines the system's performance quantitatively and provides example harmonisations generated by the system. In section 5 we compare our system to related work, and in section 6 we suggest some possible enhancements.

## 2 Musical Background

Since the sixteenth century, the music of the Lutheran church had been centred on the 'chorale'. Chorales were hymns, poetic words set to music: a famous early example is Martin Luther's "Ein' feste Burg ist unser Gott". At first chorales had only relatively simple melodic lines, but soon composers began to arrange more complex music to accompany the original tunes. In the pieces by Bach which we use here, the chorale tune is taken generally unchanged in the highest voice, and three other musical parts are created alongside it, supporting it and each other. By the eighteenth century, a complex system of rules had developed, dictating what combinations of notes should be played at the same time or following previous notes. The added lines of music should not fit too easily with the melody, but should not clash with it too much either. Dissonance can improve the music, if it is resolved into a pleasant consonance.

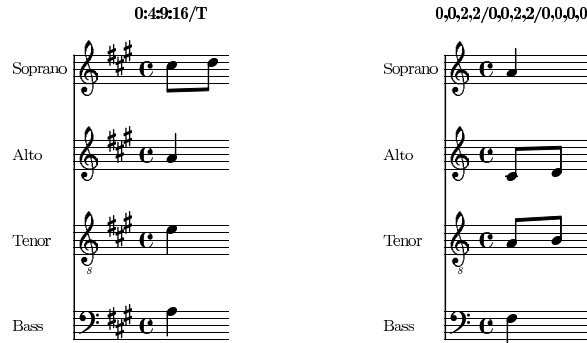

Figure 1: Hidden state representations (a) for harmonisation, (b) for ornamentation.

The training and test chorales used here are divided into two sets: one for chorales in 'major' keys, and one for chorales in 'minor' keys. Major and minor keys are based around different sets of notes, and musical lines in major and minor keys behave differently.

The representation we use to model harmonisations divides up chorales into discrete time-steps according to the regular beat underlying their musical rhythm. At each time-step we represent the notes in the various musical parts by counting how far apart they are in terms of all the possible 'semitone' notes.

# 3 Harmonisation Model

## 3.1 HMM for Harmonisation

We construct a Hidden Markov model in which the visible states are melody notes and the hidden states are chords. A sequence of observed events makes up a melody line, and a sequence of hidden events makes up a possible harmonisation for a melody line. We denote the sequence of melody notes as $Y$ and the harmonic motion as $C$, with $y_t$ representing the melody at time $t$, and $c_t$ the harmonic state.

Hidden Markov Models are generative models: here we model how a visible melody line is emitted by a hidden sequence of harmonies. This makes sense in musical terms, since we can view a chorale as having an underlying harmonic structure, and the individual notes of the melody line as chosen to be compatible with this harmonic state at each time step. We will create separate models for chorales in major and minor keys, since these groups have different harmonic structures.

For our model we divide each chorale into time steps of a single beat, making the assumption that the harmonic state does not change during a beat. (Typically there are three or four beats in a bar.) We want to create a model which we can use to predict three further notes at each of these time steps, one for each of the three additional musical lines in the harmonisation.

There are many possible hidden state representations from which to choose. Here we represent a choice of notes by a list of pitch intervals. By using intervals in this way we represent the relationship between the added notes and the melody at a given time step, without reference to the absolute pitch of the melody note. These interval sets alone would be harmonically ambiguous, so we disambiguate them using harmonic labels, which are included in the training data set. Adding harmonic labels means that our hidden symbols not only identify a particular chord, but also the harmonic function that the chord is serving. Figure 1(a) shows the representation used for some example notes. Here (an A major

chord) the alto, tenor and bass notes are respectively 4, 9, and 16 semitones below the soprano melody. The harmonic label is 'T', labelling this as functionally a 'tonic' chord. Our representation of both melody and harmony distinguishes between a note which is continued from the previous beat and a repeated note.

We make a first-order Markov assumption concerning the transition probabilities between the hidden states, which represent choices of chord on an individual beat:

$$P(c_t|c_{t-1}, c_{t-2}, \ldots, c_0) = P(c_t|c_{t-1}).$$

We make a similar assumption concerning emission probabilities to model how the observed event, a melody note, results from the hidden state, a chord:

$$P(y_t|c_t, \ldots, c_0, y_{t-1}, \ldots, y_0) = P(y_t|c_t).$$

In the Hidden Markov Models used here, the 'hidden' states of chords and harmonic symbols are in fact visible in the data during training. This means that we can learn transition and emission probabilities directly from observations in our training data set of harmonisations. We use additive smoothing (adding 0.01 to each bin) to deal with zero counts in the training data.

Using a Hidden Markov Model framework allows us to conduct efficient inference over our harmonisation choices. In this way our harmonisation system will 'plan' over an entire harmonisation rather than simply making immediate choices based on the local context. This means, for example, that we can hope to compose appropriate 'cadences' to bring our harmonisations to pleasant closes rather than finishing abruptly.

Given a new melody line, we can use the Viterbi algorithm to find the most likely state sequence, and thus harmonisation, given our model. We can also provide alternative harmonisations by sampling from the posterior [see 1, p. 156], as explained below.

### 3.2 Sampling Alternative Harmonisations

Using $\alpha_{t-1}(j)$, the probability of seeing the observed events of a sequence up to time $t-1$ and finishing in state $j$, we can calculate the probability of seeing the first $t-1$ events, finishing in state $j$, and then transitioning to state $k$ at the next step:

$$P(y_0, y_1, \ldots, y_{t-1}, c_{t-1} = j, c_t = k) = \alpha_{t-1}(j)P(c_t = k|c_{t-1} = j).$$

We can use this to calculate $\rho_t(j|k)$, the probability that we are in state $j$ at time $t-1$ given the observed events up to time $t-1$, and given that we will be in state $k$ at time $t$:

$$\rho_t(j|k) = P(c_{t-1} = j|y_0, y_1, \ldots, y_{t-1}, c_t = k) = \frac{\alpha_{t-1}(j)P(c_t = k|c_{t-1} = j)}{\sum_l \alpha_{t-1}(l)P(c_t = k|c_{t-1} = l)}.$$

To sample from $P(C|Y)$ we first choose the final state by sampling from its probability distribution according to the model:

$$P(c_T = j|y_0, y_1, \ldots, y_T) = \frac{\alpha_T(j)}{\sum_l \alpha_T(l)}.$$

Once we have chosen a value for the final state $c_T$, we can use the variables $\rho_t(j|k)$ to sample backwards through the sequence:

$$P(c_t = j|y_0, y_1, \ldots, y_T, c_{t+1}) = \rho_{t+1}(j|c_{t+1}).$$

### 3.3 HMM for Ornamentation

The chorale harmonisations produced by the Hidden Markov Model described above harmonise the original melody according to beat-long time steps. Chorale harmonisations are

Table 1: Comparison of predictive power achieved by different models of harmonic sequences on training and test data sets (nats).

|  | Training (maj) | Test (maj) | Training (min) | Training (min) |
|---|---|---|---|---|
| $-\frac{1}{T}\ln P(C\|Y)$ | 2.56 | 4.90 | 2.66 | 5.02 |
| $-\frac{1}{T}\sum \ln P(c_t\|y_t)$ | 3.00 | 3.22 | 3.52 | 4.33 |
| $-\frac{1}{T}\sum \ln P(c_t\|c_{t-1})$ | 5.41 | 7.08 | 5.50 | 7.21 |
| $-\frac{1}{T}\sum \ln P(c_t)$ | 6.43 | 7.61 | 6.57 | 7.84 |

not limited to this rhythmic form, so here we add a secondary ornamentation stage which can add passing notes to decorate these harmonisations. Generating a harmonisation and adding the ornamentation as a second stage greatly reduces the number of hidden states in the initial harmonisation model: if we went straight to fully-ornamented hidden states then the data available to us concerning each state would be extremely limited. Moreover, since the passing notes do not change the harmonic structure of a piece but only ornament it, adding these passing notes after first determining the harmonic structure for a chorale is a plausible compositional process.

We conduct ornamentation by means of a second Hidden Markov Model. The notes added in this ornamentation stage generally smooth out the movement between notes in a line of music, so we set up the visible states in terms of how much the three harmonising musical lines rise or fall from one time-step to the next. The hidden states describe ornamentation of this motion in terms of the movement made by each part during the time step, relative to its starting pitch. This relative motion is described at a time resolution four times as fine as the harmonic movement. On the first of the four quarter-beats we always leave notes as they were, so we have to make predictions only for the final three quarter-beats. Figure 1(b) shows an example of the representation used. In this example, the alto and tenor lines remain at the same pitch for the second quarter-beat as they were for the first, and rise by two semitones for the third and fourth quarter-beats, so are both represented as '0,0,2,2', while the bass line does not change pitch at all, so is represented as '0,0,0,0'.

## 4 Results

Our training and test data are derived from chorale harmonisations by Johann Sebastian Bach.[1] These provide a relatively large set of harmonisations by a single composer, and are long established as a standard reference among music theorists. There are 202 chorales in major keys of which 121 were used for training and 81 used for testing; and 180 chorales in minor keys (split 108/72).

Using a probabilistic framework allows us to give quantitative answers to questions about the performance of the harmonisation system. There are many quantities we could compute, but here we will look at how high a probability the model assigns to Bach's own harmonisations given the respective melody lines. We calculate average negative log probabilities per symbol, which describe how predictable the symbols are under the model. These quantities provide sample estimates of cross-entropy. Whereas verbal descriptions of harmonisation performance are unavoidably vague and hard to compare, these figures allow our model's performance to be directly compared with that of any future probabilistic harmonisation system.

Table 1 shows the average negative log probability per symbol of Bach's chord symbol

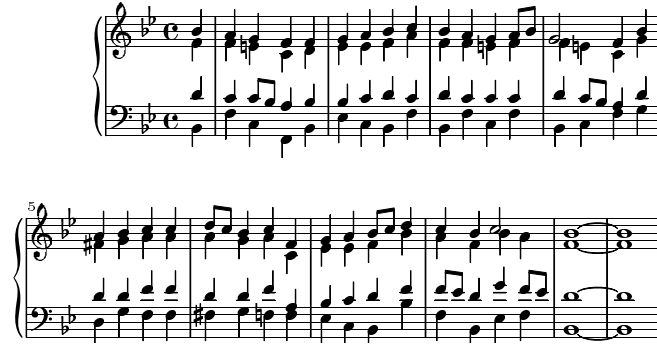

Figure 2: Most likely harmonisation under our model of chorale K4, BWV 48

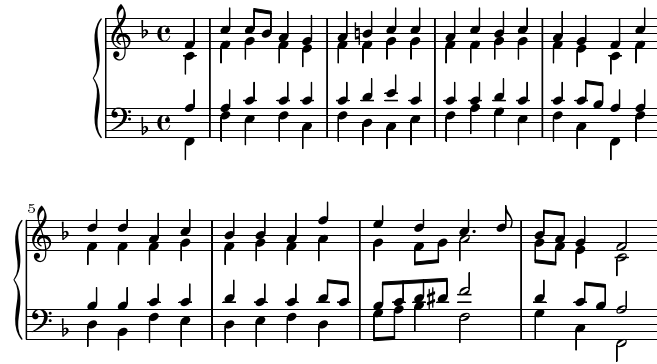

Figure 3: Most likely harmonisation under our model of chorale K389, BWV 438

sequences given their respective melodic symbol sequences, $-\frac{1}{T}\ln P(C|Y)$, on training and test data sets of chorales in major and minor keys. As a comparison we give analogous negative log probabilities for a model predicting chord states from their respective melody notes, $-\frac{1}{T}\sum \ln P(c_t|y_t)$, for a simple Markov chain between the chord states, $-\frac{1}{T}\sum \ln P(c_t|c_{t-1})$, and for a model which assumes that the chord states are independently drawn, $-\frac{1}{T}\sum \ln P(c_t)$. The Hidden Markov Model here has 5046 hidden chord states and 58 visible melody states.

The Hidden Markov Model finds a better fit to the training data than the simpler models: to choose a good chord for a particular beat we need to take into account both the melody note on that beat and the surrounding chords. Even the simplest model of the data, which assumes that each chord is drawn independently, performs worse on the test data than the training data, showing that we are suffering from sparse data. There are many chords, chord to melody note emissions, and especially chord to chord transitions, that are seen in the test data but never occur in the training data. The models' performance with unseen data could be improved by using a more sophisticated smoothing method, for example taking into account the overall relative frequencies of harmonic symbols when assigning probabilities to unseen chord transitions. However, this lower performance with unseen test data is not a problem for the task we approach here, of generating new harmonisations, as long as we can learn a large enough vocabulary of events from the training data to be able to find good harmonisations for new chorale melodies.

Figures 2 and 3 show the most likely harmonisations under our model for two short

chorales. The system has generated reasonable harmonisations. We can see, for example, passages of parallel and contrary motion between the different parts. There is an appropriate harmonic movement through the harmonisations, and they come to plausible cadences.

The generated harmonisations suffer somewhat from not taking into account the flow of the individual musical lines which we add. There are large jumps, especially in the bass line, more often than is desirable – the bass line suffers most since has the greatest variance with respect to the soprano melody. This excessive jumping also feeds through to reduce the performance of the ornamentation stage, creating visible states which are unseen in the training data. The model structure means that the most likely harmonisation leaves these states unornamented. Nevertheless, where ornamentation has been added it fits with its context and enhances the harmonisations.

The authors will publish further example harmonisations, including MIDI files, online at `http://www.tardis.ed.ac.uk/~moray/harmony/`.

## 5  Relationship to previous work

Even while Bach was still composing chorales, music theorists were catching up with musical practice by writing treatises to explain and to teach harmonisation. Two famous examples, Rameau's *Treatise on Harmony* [2] and the *Gradus ad Parnassum* by Fux [3], show how musical style was systematised and formalised into sets of rules. The traditional formulation of harmonisation technique in terms of rules suggests that we might create an automatic harmonisation system by finding as many rules as we can and encoding them as a consistent set of constraints. Pachet and Roy [4] provide a good overview of constraint-based harmonisation systems. For example, one early system [5] takes rules from Fux and assigns penalties according to the seriousness of each rule being broken. This system then conducts a modified best-first search to produce harmonisations. Using standard constraint-satisfaction techniques for harmonisation is problematic, since the space and time needs of the solver tend to rise extremely quickly with the length of the piece.

Several systems have applied genetic programming techniques to harmonisation, for example McIntyre [6]. These are similar to the constraint-based systems described above, but instead of using hard constraints they encode their rules as a fitness function, and try to optimise that function by evolutionary techniques. Phon-Amnuaisuk and Wiggins [7] are reserved in their assessment of genetic programming for harmonisation. They make a direct comparison with an ordinary constraint-based system, and conclude that the performance of each system is related to the amount of knowledge encoded in it rather than the particular technique it uses. In their comparison the ordinary constraint-based system actually performs much better, and they argue that this is because it possesses implicit control knowledge which the system based on the genetic algorithm lacks.

Even if they can be made more efficient, these rule-based systems do not perform the full task of our harmonisation system. They take a large set of rules written by a human and attempt to find a valid solution, whereas our system learns its rules from examples.

Hild et al. [8] use neural networks to harmonise chorales. Like the Hidden Markov Models in our system, these neural networks are trained using example harmonisations. However, while two of their three subtasks use only neural networks trained on example harmonisations, their second subtask, where chords are chosen to instantiate more general harmonies, includes constraint satisfaction. Rules written by a human penalise undesirable combinations of notes, so that they will be filtered out when the best chord is chosen from all those compatible with the harmony already decided. In contrast, our model learns all its harmonic 'rules' from its training data.

Ponsford et al. [9] use $n$-gram Markov models to generate harmonic structures. Unlike in

chorale harmonisation, there is no predetermined tune with which the harmonies need to fit. The data set they use is a selection of 84 saraband dances, by 15 different seventeen-century French composers. An automatically annotated corpus is used to train Markov models using contexts of different lengths, and the weighted sum of the probabilities assigned by these models used to predict harmonic movement. Ponsford et al. create new pieces first by random generation from their models, and secondly by selecting those randomly-generated pieces which match a given template. Using templates gives better results, but the great majority of randomly-generated pieces will not match the template and so will have to be discarded. Using a Hidden Markov Model rather than simple $n$-grams allows this kind of template to be included in the model as the visible state of the system: the chorale tunes in our system can be thought of as complex templates for harmonisations. Ponsford et al. note that even with their longest context length, the cadences are poor. In our system the 'planning' ability of Hidden Markov Models, using the combination of chords and harmonic labels encoded in the hidden states, produces cadences which bring the chorale tunes to harmonic closure.

This paper stems from work described in the first author's MSc thesis [10] carried out in 2002. We have recently become aware that similar work has been carried out independently in Japan by a team led by Prof S. Sagayama [11, 12]. To our knowledge this work has been published only in Japanese[2]. The basic frameworks are similar, but there are several differences. First, their system only describes the harmonisation in terms of the harmonic label (e.g. T for tonic) and does not fully specify the voicing of the three harmony lines or ornamentation. Secondly, they do not give a quantitative evaluation of the harmonisations produced as in our Table 1. Thirdly, in [12] a Markov model on *blocks* of chord sequences rather than on individual chords is explored.

## 6 Discussion

Using the framework of probabilistic influence allows us to perform efficient inference to generate new chorale harmonisations, avoiding the computational scaling problems suffered by constraint-based harmonisation systems. We described above neural network and genetic algorithm techniques which were less compute-intensive than straightforward constraint satisfaction, but the harmonisation systems using these techniques retain a pre-programmed knowledge base, whereas our model is able to learn its harmonisation constraints from training data.

Different forms of graphical model would allow us to take into account more of the dependencies in harmonisation. For example, we could use a higher-order Markov structure, although this by itself would be likely to greatly increase the problems already seen here with sparse data. An alternative might be to use an Autoregressive Hidden Markov Model [13], which models the transitions between visible states as well as the hidden state transitions modelled by an ordinary Hidden Markov Model.

Not all of Bach's chorale harmonisations are in the same style. Some of his harmonisations are intentionally complex, and others intentionally simple. We could improve our harmonisations by modelling this stylistic variation, either manually annotating training chorales according to their style or by training a mixture of HMMs.

As we only wish to model the hidden harmonic state given the melody, rather than construct a full generative model of the data, Conditional Random Fields (CRFs) [14] provide a related but alternative framework. However, note that training such models (e.g. using iterative scaling methods) is more difficult than the simple counting methods that can be applied to the HMM case. On the other hand the use of the CRF framework would have

some advantages, in that additional features could be incorporated. For example, we might be able to make better predictions by taking into account the current time step's position within its musical bar. Music theory recognises a hierarchy of stressed beats within a bar, and harmonic movement should correlated with these stresses. The ornamentation process especially might benefit from a feature-based approach.

Our system described above only considers chords as sets of intervals, and thus does not have a notion of the key of a piece (other than major or minor). However, voices have a preferred range and thus the notes that should be used do depend on the key, so the key signature could also be used as a feature in a CRF. Taking into account the natural range of each voice would prevent the bass line from descending too low and keep the three parts closer together. In general more interesting harmonies result when musical lines are closer together and their movements are more constrained. Another dimension that could be explored with CRFs would be to take into account the words of the chorales, since Bach's own harmonisations are affected by the properties of the texts as well as of the melodies.

## Acknowledgments

MA gratefully acknowledges support through a research studentship from Microsoft Research Ltd.

## Footnotes

[1] We used a computer-readable edition of Bach's chorales downloaded from `ftp://i11ftp.ira.uka.de/pub/neuro/dominik/midifiles/bach.zip`

[2]We thank Yoshinori Shiga for explaining this work to us.

## References

[1] R. Durbin, S. R. Eddy, A. Krogh, and G. Mitchison. *Biological sequence analysis*. Cambridge University Press, 1998.

[2] J.-P. Rameau. *Traité de l'Harmonie reduite à ses principes naturels*. Paris, 1722.

[3] J. J. Fux. *Gradus ad Parnassum*. Vienna, 1725.

[4] F. Pachet and P. Roy. Musical harmonization with constraints: A survey. *Constraints*, 6(1): 7–19, 2001.

[5] B. Schottstaedt. Automatic species counterpoint. Technical report, Stanford University CCRMA, 1989.

[6] R. A. McIntyre. Bach in a box: The evolution of four-part baroque harmony using the genetic algorithm. In *Proceedings of the IEEE Conference on Evolutionary Computation*, 1994.

[7] S. Phon-Amnuaisuk and G. A. Wiggins. The four-part harmonisation problem: a comparison between genetic algorithms and a rule-based system. In *Proceedings of the AISB'99 Symposium on Musical Creativity*, 1999.

[8] H. Hild, J. Feulner, and W. Menzel. HARMONET: A neural net for harmonizing chorales in the style of J.S. Bach. In R.P. Lippman, J.E. Moody, and D.S. Touretzky, editors, *Advances in Neural Information Processing 4*, pages 267–274. Morgan Kaufmann, 1992.

[9] D. Ponsford, G. Wiggins, and C. Mellish. Statistical learning of harmonic movement. *Journal of New Music Research*, 1999.

[10] M. M. Allan. Harmonising Chorales in the Style of Johann Sebastian Bach. Master's thesis, School of Informatics, University of Edinburgh, 2002.

[11] T. Kawakami. Hidden Markov Model for Automatic Harmonization of Given Melodies. Master's thesis, School of Information Science, JAIST, 2000. In Japanese.

[12] K. Sugawara, T. Nishimoto, and S. Sagayama. Automatic harmonization for melodies based on HMMs including note-chain probability. Technical Report 2003-MUS-53, Acoustic Society of Japan, December 2003. In Japanese.

[13] P. C. Woodland. Hidden Markov Models using vector linear prediction and discriminative output distributions. In *Proc ICASSP*, volume I, pages 509–512, 1992.

[14] J. D. Lafferty, A. McCallum, and F. C. N. Pereira. Conditional Random Fields: probabilistic models for segmenting and labeling sequence data. In *Proc ICML*, pages 282–289, 2001.
